# Configuration Estimates Improve Pedestrian Finding

**Duan Tran**[*]
U.Illinois at Urbana-Champaign
Urbana, IL 61801 USA
ddtran2@uiuc.edu

**D.A. Forsyth**
U.Illinois at Urbana-Champaign
Urbana, IL 61801 USA
daf@uiuc.edu

## Abstract

Fair discriminative pedestrian finders are now available. In fact, these pedestrian finders make most errors on pedestrians in configurations that are uncommon in the training data, for example, mounting a bicycle. This is undesirable. However, the human configuration can itself be estimated discriminatively using structure learning. We demonstrate a pedestrian finder which first finds the most likely human pose in the window using a discriminative procedure trained with structure learning on a small dataset. We then present features (local histogram of oriented gradient and local PCA of gradient) based on that configuration to an SVM classifier. We show, using the INRIA Person dataset, that estimates of configuration significantly improve the accuracy of a discriminative pedestrian finder.

## 1 Introduction

Very accurate pedestrian detectors are an important technical goal; approximately half-a-million pedestrians are killed by cars each year (1997 figures, in [1]). At relatively low resolution, pedestrians tend to have a characteristic appearance. Generally, one must cope with lateral or frontal views of a walk. In these cases, one will see either a "lollipop" shape — the torso is wider than the legs, which are together in the stance phase of the walk — or a "scissor" shape — where the legs are swinging in the walk. This encourages the use of template matching. Early template matchers include: support vector machines applied to a wavelet expansion ([2], and variants described in [3]); a neural network applied to stereoscopic reconstructions [4]; chamfer matching to a hierachy of contour templates [5]; a likelihood threshold applied to a random field model [6]; an SVM applied to spatial wavelets stacked over four frames to give dynamical cues [3]; a cascade architecture applied to spatial averages of temporal differences [7]; and a temporal version of chamfer matching to a hierachy of contour templates [8].

By far one of the most successful static template matcher is due to Dalal and Triggs [9]. Their method is based on a comprehensive study of features and their effects on performance for the pedestrian detection problem. The method that performs best involves a histogram of oriented gradient responses (a **HOG** descriptor). This is a variant of Lowe's SIFT feature [10]. Each window is decomposed into overlapping blocks (large spatial domains) of cells (smaller spatial domains).In each block, a histogram of gradient directions (or edge orientations) is computed for each cell with a measure of histogram "energy". These cell histograms are concatenated into block histograms followed by normalization which obtains a modicum of illumination invariance. The detection window is tiled with an overlapping grid. Within each block HOG descriptors are computed, and the

---

[*]We would like to thank Alexander Sorokin for his providing the annotation software and Pietro Perona for insightful comments. This work was supported by Vietnam Education Foundation as well as in part by the National Science Foundation under IIS - 0534837 and in part by the Office of Naval Research under N00014-01-1-0890 as part of the MURI program. Any opinions, findings and conclusions or recommendations expressed in this material are those of the author(s) and do not necessarily reflect those of the National Science Foundation or the Office of Naval Research.

resulting feature vector is presented to an SVM. Dalal and Triggs show this method produces no errors on the 709 image MIT dataset of [2]; they describe an expanded dataset of 1805 images. Furthermore, they compare HOG descriptors with the original method of Papageorgiou and Poggio [2]; with an extended version of the Haar wavelets of Mohan *et al.* [11]; with the PCA-Sift of Ke and Sukthankar ([12]; see also [13]); and with the shape contexts of Belongie *et al.* [14]. The HOG descriptors outperform all other methods. Recently, Sabzmeydani and Mori [15] reported improved results by using AdaBoost to select *shapelet* features (triplets of location, direction and strength of local average gradient responses in different directions).

A key difficulty with pedestrian detection is that detectors must work on human configurations not often seen in datasets. For systems to be useful, they cannot fail even on configurations that are very uncommon — it is not acceptable to run people over when they stand on their hands. There is some evidence (figure 1) that less common configurations present real difficulties for very good current pedestrian detectors (our reimplementation of Dalal and Triggs' work [9]).

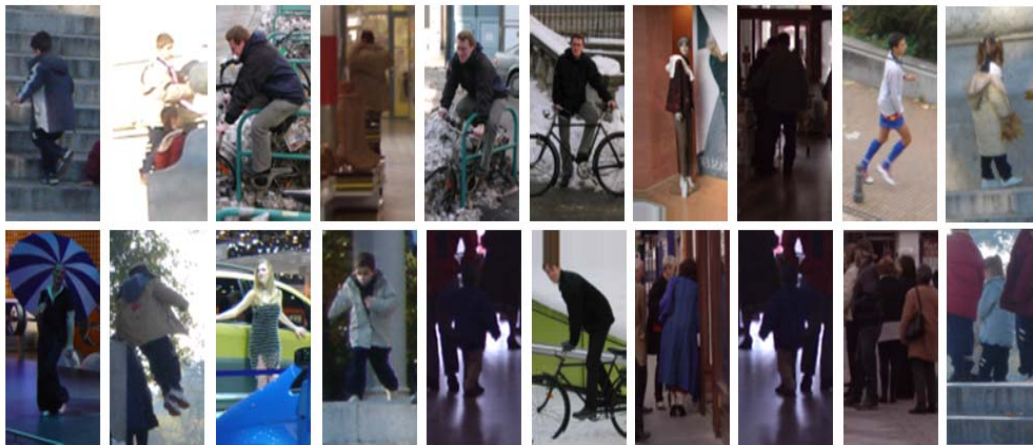

**Figure 1.** *Configuration estimates result in our method producing fewer false negatives than our implementation of Dalal and Triggs does. The figure shows typical images which are incorrectly classified by our implementation of Dalal and Triggs, but correctly classified when a configuration estimate is attached. We conjecture that a configuration estimate can avoid problems with occlusion or contrast failure because the configuration estimate reduces noise and the detector can use lower detection thresholds.*

## 1.1 Configuration and Parts

Detecting pedestrians with templates most likely works because pedestrians appear in a relatively limited range of configurations and views (e.g. "Our HOG detectors cue mainly on silhouette contours (especially the head, shoulders and feet)" [9], p.893). It appears certain that using the architecture of constructing features for whole image windows and then throwing the result into a classifier could be used to build a person-finder for arbitrary configurations and arbitrary views only with a major engineering effort. The set of examples required would be spectacularly large, for example. This is unattractive, because this set of examples implicitly encodes a set of facts that are relatively easy to make explicit. In particular, people are made of body segments which individually have a quite simple structure, and these segments are connected into a kinematic structure which is quite well understood.

All this suggests finding people by finding the parts and then reasoning about their layout — essentially, building templates with complex internal kinematics. The core idea is very old (see the review in [16]) but the details are hard to get right and important novel formulations are a regular feature of the current research literature.

Simply identifying the body parts can be hard. **Discriminative approaches** use classifiers to detect parts, then reason about configuration [11]. **Generative approaches** compare predictions of part appearance with the image; one can use a tree structured configuration model [17], or an arbitrary graph [18]. If one has a video sequence, part appearance can itself be learned [19, 20]; more recently,

Ramanan has shown knowledge of articulation properties gives an appearance model in a single image [21]. **Mixed approaches** use a discriminative model to identify parts, then a generative model to construct and evaluate assemblies [22, 23, 24]. **Codebook approaches** avoid explicitly modelling body segments, and instead use unsupervised methods to find part decompositions that are good for recognition (rather than disarticulation) [25].

Our pedestrian detection strategy consists of two steps: first, for each window, we estimate the configuration of the best person available in that window; second, we extract features for that window conditioned on the configuration estimate, and pass these features to a support vector machine classifier, which makes the final decision on the window.

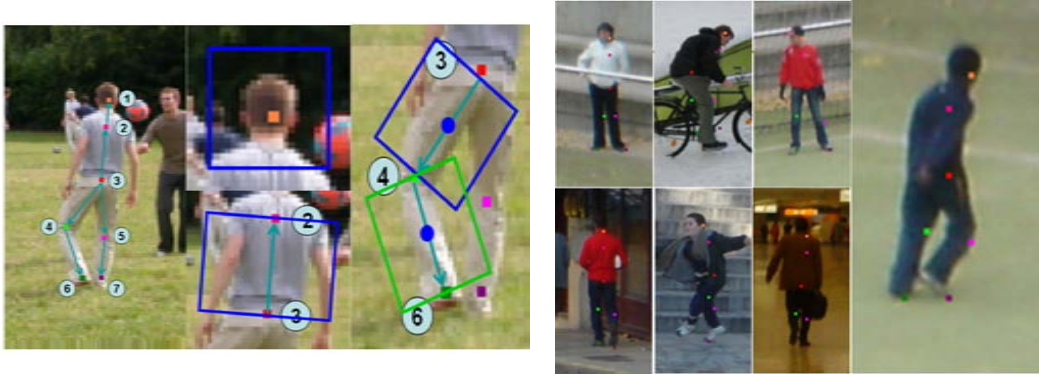

**Figure 2.** This figure is best viewed in color. *Our model of human layout is parametrized by seven vertices, shown on an example on the* **far left***. The root is at the hip; the arrows give the direction of conditional dependence. Given a set of features, the extremal model can be identified by dynamic programming on point locations. We compute segment features by placing a box around some vertices (as in the* **head***), or pairs of vertices (as in the* **torso** *and* **leg***). Histogram features are then computed for base points referred to the box coordinate frame; the histogram is shifted by the orientation of the box axis (section 3) within the rectified box. On the* **far right***, a window showing the color key for our structure learning points; dark green is a foot, green a knee, dark purple the other foot, purple the other knee, etc. Note that structure learning is capable of finding distinction of left legs (green points) and right legs (pink points). On the* **center right***, examples of configurations estimated by our configuration estimator after 20 rounds of structure learning to estimate* **W***.*

## 2   Configuration Estimation and Structure Learning

We are presented with a window within which may lie a pedestrian. We would like to be able to estimate the most likely configuration for any pedestrian present. Our research hypothesis is that this estimate will improve pedestrian detector perfomance by reducing the amount of noise the final detector must cope with — essentially, the segmentation of the pedestrian is improved from a window to a (rectified) figure. We follow convention (established by [26]) and model the configuration of a person as a tree model of segments (figure 2), with a score of segment quality and a score of segment-segment configuration. We ignore arms because they are small and difficult to localize. Our configuration estimation procedure will use dynamic programming to extract the best configuration estimate from a set of scores depending on the location of vertices on the body model.

However, we do not know which features are most effective at estimating segment location; this is a well established difficulty in the literature [16]. **Structure learning** is a method that uses a series of correct examples to estimate appropriate weightings of features relative to one another to produce a score that is effective at estimating configuration [27, 28]. We will write the image as $\mathcal{I}$; coordinates in the image as $\mathbf{x}$; the coordinates of an estimated configuration as $\mathbf{y}$ (which is a stack of 7 point coordinates); the score for this configuration as $\mathbf{W}^T\mathbf{f}(\mathcal{I}, \mathbf{x}; \mathbf{y})$ (which is a linear combination of a collection of scores, each of which depends on the configuration and the image).

For a given image $\mathcal{I}_0$ and known $\mathbf{W}$ and $\mathbf{f}$, the best configuration estimate is

$$\arg\max_{\mathbf{y} \in \mathbf{y}(\mathcal{I}_0)} \mathbf{W}^T\mathbf{f}(\mathcal{I}_0, \mathbf{x}; \mathbf{y})$$

and this can be found with dynamic programming for appropriate choice of $\mathbf{f}$ and $\mathbf{y}(\mathcal{I}_0)$. There is a variety of sensible choices of features for identifying body segments, but there is little evidence that a particular choice of features is best; different choices of $\mathbf{W}$ may lead to quite different behaviours. In particular, we will collect a wide range of features likely to identify segments well in $\mathbf{f}$, and wish to learn a choice of $\mathbf{W}$ that will give good configuration estimates.

We choose a loss function $L(\mathbf{y}_t, \mathbf{y}_p)$ that gives the cost of predicting $\mathbf{y}_p$ when the correct answer is $\mathbf{y}_t$. Write the set of n examples as $\mathcal{E}$, and $\mathbf{y}_{p,i}$ as the prediction for the $i$'th example. Structure learning must now estimate a $\mathbf{W}$ to minimize the hinge loss as in [29]

$$\frac{1}{2} \parallel \mathbf{W} \parallel^2 + \frac{1}{n} \sum_{i \in examples} \beta_i \xi_i$$

subject to the constraints

$$\forall i \in \mathcal{E}, \ \mathbf{W}^T \mathbf{f}(\mathcal{I}_i, \mathbf{x}; \mathbf{y}_{t,i}) + \xi_i \geq \max_{\mathbf{y}_{p,i} \in \mathbf{y}(\mathcal{I}_i)} (\mathbf{W}^T(\mathcal{I}_i, \mathbf{x}; \mathbf{y}_{p,i}) + L(\mathbf{y}_{t,i}, \mathbf{y}_{p,i}))$$

At the minimum, the slack variables $\xi_i$ happen at the equality of the constraints. Therefore, we can move the constraints to the objective function, which is:

$$\frac{1}{2} \parallel \mathbf{W} \parallel^2 + \frac{1}{n} \sum_{i \in examples} \beta_i \big( \max_{\mathbf{y}_{p,i} \in \mathbf{y}(\mathcal{I}_i)} (\mathbf{W}^T(\mathcal{I}_i, \mathbf{x}; \mathbf{y}_{p,i}) + L(\mathbf{y}_{t,i}, \mathbf{y}_{p,i})) - \mathbf{W}^T \mathbf{f}(\mathcal{I}_i, \mathbf{x}; \mathbf{y}_{t,i}) \big)$$

Notice that this function is convex, but not differentiable. We follow Ratliff *et al.* [29], and use the subgradient method (see [30]) to minimize. In this case, the derivative of the cost function at an extremal $\mathbf{y}_{p,i}$ is a subgradient (but not a gradient, because the cost function is not differentiable everywhere).

## 3 Features

There are two sets of features: first, those used for estimating configuration of a person from a window; and second, those used to determine whether a person is present conditioned on the best estimate of configuration.

### 3.1 Features for Estimating Configuration

We use a tree structured model, given in figure 2. The tree is given by the position of seven points, and encodes the head, torso and legs; arms are excluded because they are small and difficult to identify, and pedestrians can be identified without localizing arms. The tree is rooted at hips, and the arrows give the direction of conditional dependence. We assume that $torso, leftleg, rightleg$ are conditionally independent given the root (at the hip).

The feature vector $\mathbf{f}(\mathcal{I}, \mathbf{x}; \mathbf{y})$ contains two types of feature: appearance features encode the appearance of putative segments; and geometric features encode relative and absolute configuration of the body segments.

Each **geometric feature** depends on at most three point positions. We use three types of feature. First, the length of a segment, represented as a 15-dimensional binary vector whose elements encode whether the segment is longer than each of a set of test segments. Second, the cosine of the angle between a segment and the vertical axis. Third, the cosine of the angle between pairs of adjoining segments (except at the lower torso, for complexity reasons); this allows the structure learning method to prefer straight backs, and reasonable knees.

**Appearance features** are computed for rectangles constructed from pairs of points adjacent in the tree. For each rectangle, we compute Histogram of Oriented Gradient (HOG) features, after [9]. These features have a strong record in pedestrian detection, because they can detect the patterns of orientation associated with characteristic segment outlines (typically, strong vertical orientations in the frame of the segment for torso and legs; strong horizontal orientations at the shoulders and head). However, histograms involve spatial pooling; this means that one can have many strong vertical orientations that do not join up to form a segment boundary. This effect means that HOG features alone are not particularly effective at estimating configuration.

To counter this effect, we use the local gradient features described by Ke and Sukthankar [12]. To form these features, we concatenate the horizontal and vertical gradients of the patches in the segment coordinate frame, then normalize and apply PCA to reduce the number of dimensions. Since we want to model the appearance, we do not align the orientation to a canonical orientation as in PCA-SIFT. This feature reveals whether the pattern of a body part appears at that location. The PCA space for each body part is constructed from 500 annotated positive examples.

## 3.2 Features for Detection

Once the best configuration has been obtained for a window, we must determine whether a person is present or not. We do this with a support vector machine. Generally, the features that determine configuration should also be good for determining whether a person is present or not. However, a set of HOG features for the whole image window has been shown to be good at pedestrian detection [9]. The support vector machine should be able to distinguish between good and bad features, so it is natural to concatenate the configuration features described above with a set of HOG features. We find it helpful to reduce the dimension of the set of HOG features to 500, using principal components. We find that these whole window features help recover from incorrect structure predictions. These combined features are used in training the SVM classifier and in detection as well.

## 4 Results

**Dataset:** We use INRIA Person, consisting of 2416 pedestrian images (1208 images with their left-right reflections) and 1218 background images for training. For testing, there are 1126 pedestrian images (563 images with their left-right reflections) and 453 background images.

**Training structure learning:** we manually annotate 500 selected pedestrian images in the training set examples. We use all 500 annotated examples to build the PCA spaces for each body segment. In training, each example is learned to update the weight vector. The order of selecting examples in each round is randomly drawn based on the differences of their scores on the predictions and their scores on the true targets. For each round, we choose 300 examples drawn (since structure learning is expensive). We have trained the structure learning on 10 rounds and 20 rounds for comparisons.

**Quality of configuration estimates:** Configuration estimates look good (figure 2). A persistent nuisance associated with pictorial structure models of people is the tendency of such models to place legs on top of one another. This occurs if one uses only appearance and relative geometric features. However, our results suggest that if one uses absolute configuration features as well as different appearance features for left and right legs (implicit in the structure learning procedure), the left and right legs are identified correctly. The conditional independence assumption (which means we cannot use the angle between the legs as a feature) does not appear to cause problems, perhaps because absolute configuration features are sufficient.

**Bootstrapping the SVM:** The final SVM is bootstrapped, as in [9]. We use 2146 pedestrian images with 2756 window images extracted from 1218 background images. We apply the learned structure model to generate on these 2416 positive examples and 2756 negative examples to train the initial SVM classifier. We then use this classifier to scan over 1218 background images with step side of 32 pixels and find hard examples (including false positives and true negatives of low confidence by using LibSVM [31] with probability option). These negatives yield a bootstrap training set for the final SVM classifier. This bootstrap learning helps to reduce the false alarm significantly.

**Testing:** We test on 1126 positive images and scan 64x128 image windows over 453 negative test images, stepping by 16 pixels, a total of 182, 934 negative windows.

**Scanning rate and comparison:** Pedestrian detection systems work by scanning image windows, and presenting each window to a detector. Dalal and Triggs established a methodology for evaluating pedestrian detectors, which is now quite widely used. Their dataset offers a set of positive *windows* (where pedestrians are centered), and a set of negative images. The negative images produce a pool of negative *windows*, and the detector is evaluated on detect rate on the positive windows and the false positive per window (FPPW) rate on the negative windows. This strategy — which evaluates the detector, rather than the combination of detection and scanning — is appropriate for comparing systems that scan image windows at approximately the same high rate. Current systems

do so, because the detectors require nearly centered pedestrians. However, the important practical parameter for evaluating a system is the false positive per image (FPPI) rate. If one has a detector that does not require a pedestrian to be centered in the image window, then one can obtain the same detect rate while scanning fewer image windows. In turn, the FPPI rate will go down even if the FPPW rate is fixed. To date, this issue has not arisen, because pedestrian detectors have required pedestrians to be centered.

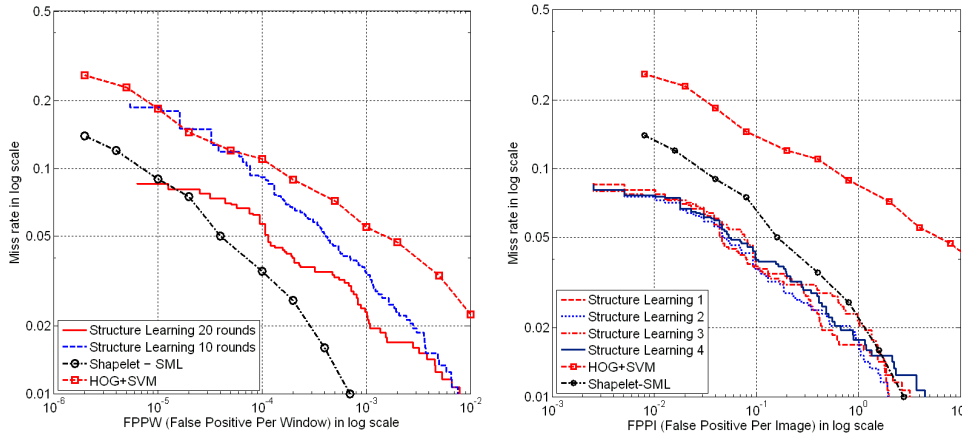

**Figure 3.** **Left:** *a comparison of our method with the best detector of Dalal and Triggs, and the detector of Sabzmaydani and Mori,* on the basis of FPPW rate*. This comparison ignores the fact that we can look at fewer image windows without loss of system sensitivity. We show ROC's for a configuration estimator trained on 10 (blue) and 20 (red) rounds of structure learning. With 20 rounds of structure learning, our detector easily outperforms that of Dalal and Triggs. Note that at high specificity, our detector is slightly more sensitive than that of Sabzmaydani and Mori, too.* **Right:** *a comparison of our method with the best detector of Dalal and Triggs, and the detector of Sabzmaydani and Mori,* on the basis of FPPI rate*. This comparison takes into account the fact that we can look at fewer image windows (by a factor of four). However, scanning by larger steps might cause a loss of sensitivity. We test this with a procedure of replicating positive examples, described in the text, and show the results of four runs. The low variance in the detect rate under this procedure shows that our detector is highly insensitive to the configuration of the pedestrian within a window. If one evaluates on the basis of false positives per image — which is likely the most important practical parameter — our system easily outperforms the state of the art.*

## 4.1  The Effect of Configuration Estimates

Figure 3 compares our detector with that of Dalal and Triggs, and of Sabzmeydani and Mori on the basis of detect and FPPW rates. We plot detect rate against FPPW rate for the three detectors. For this plot, note that at low FPPW rate our method is somewhat more sensitive than that of Sabzmeydani and Mori, but has no advantage at higher FPPW rates.

However, this does not tell the whole story. We scan images at steps of 16 pixels (rather than 8 pixels for Dalal and Triggs and Sabzmeydani and Mori). This means that we scan four times fewer windows than they do. If we can establish that the detect rate is not significantly affected by big offsets in pedestrian position, then we expect a large advantage in FPPI rate.

We evaluate the effect on the detect rate of scanning by large steps by a process of sampling. Each positive example is replaced by a total of 256 replicates, obtained by offsetting the image window by steps in the range -7 to 8 in $x$ and $y$ (figure 4). We now conduct multiple evaluation runs. For each, we select one replicate of each positive example uniformly at random. For each run, we evaluate the detect rate. A tendency of the detector to require centered pedestrians would appear as variance in the reported detect rate. The FPPI rate of the detector is not affected by this procedure, which evaluates only the spatial tuning of the detector.

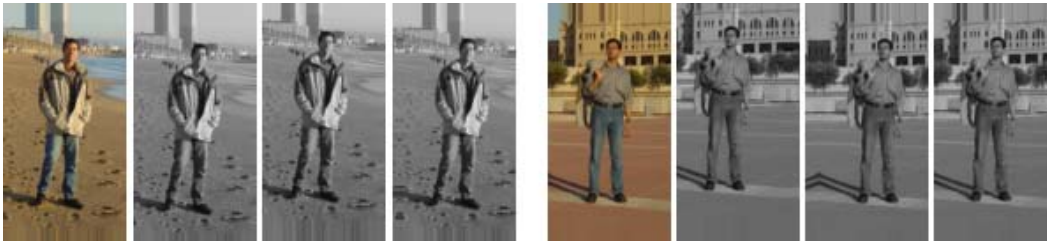

**Figure 4.** *In color, original positive examples from the INRIA test set; next to each, are three of the replicates we use to determine the effect on our detection system of scanning relatively few windows, or, equivalently, the effect on our detector of not having a pedestrian centered in the window. See section 4.1, and figure 3.*

Figure 3 compares system performance, combining detect and scanning rates, by plotting detect rate against FPPI rate. We show four evaluation runs for our system; there is no evidence of substantial variance in detect rate. Our system shows a very substantial increase in detect rate at fixed FPPI rate.

## 5  Discussion

There is a difficulty with the evaluation methodology for pedestrian detection established by Dalal and Triggs (and widely followed). A pedestrian detector that tests windows cannot find more pedestrians than there are windows. This does not usually affect the interpretation of precision and recall statistics because the windows are closely packed. However, in our method, because a pedestrian need not be centered in the window to be detected, the windows need not be closely packed, and there is a possibility of undercounting pedestrians who stand too close together. We believe that this does not occur in our current method, because our window spacing is narrow relative to the width of a pedestrian.

Part representations appear to be a natural approach to identifying people. However, to our knowledge, there is no clear evidence to date that shows compelling advantages to using such an approach (e.g. the review in [16]). We believe our method does so. Configuration estimates appear to have two important advantages. First, they result in a detector that is relatively insensitive to the placement of a pedestrian in an image window, meaning one can look at fewer image windows to obtain the same detect rate, with consequent advantages to the rate at which the system produces false positives. This is probably the dominant advantage. Second, configuration estimates appear to be a significant help at high specificity settings (notice that our method beats all others *on the FPPW criterion* at very low FPPW rates). This is most likely because the process of estimating configurations focuses the detector on important image features (rather than pooling information over space). The result would be that, when there is low contrast or a strange body configuration, the detector can use a somewhat lower detection threshold for the same FPPW rate. Figure 1 shows human configurations detected by our method but not by our implementation of Dalal and Triggs; notice the predominance of either strange body configurations or low contrast. Structure learning is an attractive method to determine which features are discriminative in configuration estimation, and it produces good configuration estimates in complex images. Future work will include: tying $W$ components for legs; evaluating arm detection; and formulating strategies to employ structure learning for detecting other objects.

## References

[1] D.M. Gavrila. Sensor-based pedestrian protection. *Intelligent Transportation Systems*, pages 77–81, 2001.

[2] C. Papageorgiou and T. Poggio. A trainable system for object detection. *Int. J. Computer Vision*, 38(1):15–33, June 2000.

[3] C.P. Papageorgiou and T. Poggio. A pattern classification approach to dynamical object detection. In *Int. Conf. on Computer Vision*, pages 1223–1228, 1999.

[4] L. Zhao and C.E. Thorpe. Stereo- and neural network-based pedestrian detection. *Intelligent Transportation Systems*, 1(3):148–154, September 2000.

[5] D. Gavrila. Pedestrian detection from a moving vehicle. In *European Conference on Computer Vision*, pages II: 37–49, 2000.

[6] Y. Wu, T. Yu, and G. Hua. A statistical field model for pedestrian detection. In *IEEE Conf. on Computer Vision and Pattern Recognition*, pages I: 1023–1030, 2005.

[7] P. Viola, M.J. Jones, and D. Snow. Detecting pedestrians using patterns of motion and appearance. *Int. J. Computer Vision*, 63(2):153–161, July 2005.

[8] M. Dimitrijevic, V. Lepetit, and P. Fua. Human body pose recognition using spatio-temporal templates. In *ICCV workshop on Modeling People and Human Interaction*, 2005.

[9] N. Dalal and B. Triggs. Histograms of oriented gradients for human detection. In *IEEE Conf. on Computer Vision and Pattern Recognition*, pages I: 886–893, 2005.

[10] D.G. Lowe. Distinctive image features from scale-invariant keypoints. *Int. J. Computer Vision*, 60(2):91–110, November 2004.

[11] A. Mohan, C.P. Papageorgiou, and T. Poggio. Example-based object detection in images by components. *IEEE T. Pattern Analysis and Machine Intelligence*, 23(4):349–361, April 2001.

[12] Y. Ke and R. Sukthankar. Pca-sift: a more distinctive representation for local image descriptors. In *IEEE Conf. on Computer Vision and Pattern Recognition*, pages II: 506–513, 2004.

[13] K. Mikolajczyk and C. Schmid. A performance evaluation of local descriptors. *IEEE T. Pattern Analysis and Machine Intelligence*, 2004. accepted.

[14] Serge Belongie, Jitendra Malik, and Jan Puzicha. Shape matching and object recognition using shape contexts. *IEEE T. Pattern Analysis and Machine Intelligence*, 24(4):509–522, 2002.

[15] P. Sabzmeydani and G. Mori. Detecting pedestrians by learning shapelet features. In *CVPR*, 2007.

[16] D.A. Forsyth, O.Arikan, L. Ikemoto, J. O'Brien, and D. Ramanan. Computational studies in human motion 1: Tracking and animation. *Foundations and Trends in Computer Vision*, 2006. In press.

[17] P.F. Felzenszwalb and D.P. Huttenlocher. Pictorial structures for object recognition. *Int. J. Computer Vision*, 61(1):55–79, January 2005.

[18] M. P. Kumar, P. H. S. Torr, and A. Zisserman. Extending pictorial structures for object recognition. In *Proceedings of the British Machine Vision Conference*, 2004.

[19] Deva Ramanan, D.A. Forsyth, and A. Zisserman. Strike a pose: Tracking people by finding stylized poses. In *IEEE Conf. on Computer Vision and Pattern Recognition*, 2005.

[20] D. Ramanan and D.A. Forsyth. Using temporal coherence to build models of animals. In *Proc. ICCV*, 2003.

[21] D. Ramanan. Learning to parse images of articulated objects. In *Proc. NIPS*, 2006.

[22] R. Ronfard, C. Schmid, and B. Triggs. Learning to parse pictures of people. In *European Conference on Computer Vision*, page IV: 700 ff., 2002.

[23] K. Mikolajczyk, C. Schmid, and A. Zisserman. Human detection based on a probabilistic assembly of robust part detectors. In *European Conference on Computer Vision*, pages Vol I: 69–82, 2004.

[24] A. Micilotta, E. Ong, and R. Bowden. Detection and tracking of humans by probabilistic body part assembly. In *British Machine Vision Conference*, volume 1, pages 429–438, 2005.

[25] B. Leibe, E. Seemann, and B. Schiele. Pedestrian detection in crowded scenes. In *IEEE Conf. on Computer Vision and Pattern Recognition*, pages I: 878–885, 2005.

[26] Pedro F. Felzenszwalb and Daniel P. Huttenlocher. Efficient matching of pictorial structures. In *IEEE Conf. on Computer Vision and Pattern Recognition*, 2000.

[27] B. Taskar. *Learning Structured Prediction Models: A Large Margin Approach*. PhD thesis, Stanford University, 2004.

[28] B. Taskar, S. Lacoste-Julien, and M. Jordan. Structured prediction via the extragradient method. In *Neural Information Processing Systems Conference*, 2005.

[29] N. Ratliff, J. A. Bagnell, and M. Zinkevich. Subgradient methods for maximum margin structured learning. In *ICML 2006 Workshop on Learning in Structured Output Spaces*, 2006.

[30] N.Z. Shor. *Minimization Methods for Non-Differentiable Functions and Applications*. 1985.

[31] Chih-Chung Chang and Chih-Jen Lin. *LIBSVM: a library for support vector machines*, 2001.
